# Support Vector Classification with Input Data Uncertainty

**Jinbo Bi**
Computer-Aided Diagnosis & Therapy Group
Siemens Medical Solutions, Inc.
Malvern, PA 19355
jinbo.bi@siemens.com

**Tong Zhang**
IBM T. J. Watson Research Center
Yorktown Heights, NY 10598
tzhang@watson.ibm.com

## Abstract

This paper investigates a new learning model in which the input data is corrupted with noise. We present a general statistical framework to tackle this problem. Based on the statistical reasoning, we propose a novel formulation of support vector classification, which allows uncertainty in input data. We derive an intuitive geometric interpretation of the proposed formulation, and develop algorithms to efficiently solve it. Empirical results are included to show that the newly formed method is superior to the standard SVM for problems with noisy input.

## 1   Introduction

In the traditional formulation of supervised learning, we seek a predictor that maps input $\mathbf{x}$ to output $y$. The predictor is constructed from a set of training examples $\{(\mathbf{x}_i, y_i)\}$. A hidden underlying assumption is that errors are confined to the output $y$. That is, the input data are not corrupted with noise; or even when noise is present in the data, its effect is ignored in the learning formulation.

However, for many applications, this assumption is unrealistic. Sampling errors, modeling errors and instrument errors may preclude the possibility of knowing the input data exactly. For example, in the problem of classifying sentences from speech recognition outputs for call-routing applications, the speech recognition system may make errors so that the observed text is corrupted with noise. In image classification applications, some features may rely on image processing outputs that introduce errors. Hence classification problems based on the observed text or image features have noisy inputs. Moreover, many systems can provide estimates for the reliability of their outputs, which measure how uncertain each element of the outputs is. This confidence information, typically ignored in the traditional learning formulations, can be useful and should be considered in the learning formulation.

A plausible approach for dealing with noisy input is to use the standard learning formulation without modeling the underlying input uncertainty. If we assume that the same noise is observed both in the training data and in the test data, then the noise will cause similar effects in the training and testing phases. Based on this (non-rigorous) reasoning, one can argue that the issue of input noise may be ignored. However, we show in this paper that by modeling input uncertainty, we can obtain more accurate predictors.

## 2 Statistical models for prediction problems with uncertain input

Consider $(\mathbf{x}_i, y_i)$, where $\mathbf{x}_i$ is corrupted with noise. Let $\mathbf{x}'_i$ be the original uncorrupted input. We consider the following data generating process: first $(\mathbf{x}'_i, y_i)$ is generated according to a distribution $p(\mathbf{x}'_i, y_i | \theta)$, where $\theta$ is an unknown parameter that should be estimated from the data; next, given $(\mathbf{x}'_i, y_i)$, we assume that $\mathbf{x}_i$ is generated from $\mathbf{x}'_i$ (but independent of $y_i$) according to a distribution $p(\mathbf{x}_i | \theta', \sigma_i, \mathbf{x}'_i)$, where $\theta'$ is another possibly unknown parameter, and $\sigma_i$ is a known parameter which is our estimate of the uncertainty (e.g. variance) for $\mathbf{x}_i$. The joint probability of $(\mathbf{x}'_i, \mathbf{x}_i, y_i)$ can be written as:

$$p(\mathbf{x}'_i, \mathbf{x}_i, y_i) = p(\mathbf{x}'_i, y_i | \theta) p(\mathbf{x}_i | \theta', \sigma_i, \mathbf{x}'_i).$$

The joint probability of $(\mathbf{x}_i, y_i)$ is obtained by integrating out the unobserved quantity $\mathbf{x}'_i$:

$$p(\mathbf{x}_i, y_i) = \int p(\mathbf{x}'_i, y_i | \theta) p(\mathbf{x}_i | \theta', \sigma_i, \mathbf{x}'_i) d\mathbf{x}'_i.$$

This model can be considered as a mixture model where each mixture component corresponds to a possible true input $\mathbf{x}'_i$ not observed. In this framework, the unknown parameter $(\theta, \theta')$ can be estimated from the data using the maximum-likelihood estimate as:

$$\max_{\theta, \theta'} \sum_i \ln p(\mathbf{x}_i, y_i | \theta, \theta') = \max_{\theta, \theta'} \sum_i \ln \int p(\mathbf{x}'_i, y_i | \theta) p(\mathbf{x}_i | \theta', \sigma_i, \mathbf{x}'_i) d\mathbf{x}'_i. \qquad (1)$$

Although this is a principled approach under our data generation process, due to the integration over the unknown true input $\mathbf{x}'_i$, it often leads to a very complicated formulation which is difficult to solve. Moreover, it is not straight-forward to extend the method to non-probability formulations such as support vector machines. Therefore we shall consider an alternative that is computationally more tractable and easier to generalize. The method we employ in this paper can be regarded as an approximation to (1), often used in engineering applications as a heuristics for mixture estimation. In this method, we simply regard each $\mathbf{x}'_i$ as a parameter of the probability model, so the maximum-likelihood becomes:

$$\max_{\theta, \theta'} \sum_i \ln \sup_{\mathbf{x}'_i} [p(\mathbf{x}'_i, y_i | \theta) p(\mathbf{x}_i | \theta', \sigma_i, \mathbf{x}'_i)]. \qquad (2)$$

If our probability model is correctly specified, then (1) is the preferred formulation. However in practice, we may not know the exact $p(\mathbf{x}_i | \theta', \sigma_i, \mathbf{x}'_i)$ (for example, we may not be able to estimate the level of uncertainty $\sigma_i$ accurately). Therefore in practice, under mis-specified probability models, (1) is not necessarily always a better method.

Intuitively (1) and (2) have similar effects since large values of $p(\mathbf{x}'_i, y_i | \theta) p(\mathbf{x}_i | \theta', \sigma_i, \mathbf{x}'_i)$ dominate the summation in $\int p(\mathbf{x}'_i, y_i | \theta) p(\mathbf{x}_i | \theta', \sigma_i, \mathbf{x}'_i) d\mathbf{x}'_i$. That is, both methods prefer a parameter configuration such that the product $p(\mathbf{x}'_i, y_i | \theta) p(\mathbf{x}_i | \theta', \sigma_i, \mathbf{x}'_i)$ is large for some $\mathbf{x}'_i$. If an observation $\mathbf{x}_i$ is contaminated with large noise so that $p(\mathbf{x}_i | \theta', \sigma_i, \mathbf{x}'_i)$ has a flat shape, then we can pick a $\mathbf{x}'_i$ that is very different from $\mathbf{x}_i$ which predicts $y_i$ well. On the other hand, if an observation $\mathbf{x}_i$ is contaminated with very small noise, then (1) and (2) penalize a parameter $\theta$ such that $p(\mathbf{x}_i, y_i | \theta)$ is small. This has the effect of ignoring data that are very uncertain and relying on data that are less contaminated.

In the literature, there are two types of statistical models: generative models and discriminative models (conditional models). We focus on discriminative modeling in this paper since it usually leads to better prediction performance. In discriminative modeling, we assume that $p(\mathbf{x}'_i, y_i | \theta)$ has a form $p(\mathbf{x}'_i, y_i | \theta) = p(\mathbf{x}'_i) p(y_i | \theta, \mathbf{x}'_i)$.

As an example, we consider regression problems with Gaussian noise:

$$p(\mathbf{x}'_i, y_i | \theta) \sim p(\mathbf{x}'_i) \exp\left(-\frac{(\theta^T \mathbf{x}'_i - y_i)^2}{2\sigma^2}\right), \quad p(\mathbf{x}_i | \theta', \sigma_i, \mathbf{x}'_i) \sim \exp\left(-\frac{\|\mathbf{x}_i - \mathbf{x}'_i\|^2}{2\sigma_i^2}\right).$$

The method in (2) becomes

$$\theta = \arg\min_{\theta} \sum_i \inf_{\mathbf{x}'_i} \left[ \frac{(\theta^T \mathbf{x}'_i - y_i)^2}{2\sigma^2} + \frac{\|\mathbf{x}_i - \mathbf{x}'_i\|^2}{2\sigma_i^2} \right]. \tag{3}$$

This formulation is closely related (but not identical) to the so-called *total least squares* (TLS) method [6, 5]. The motivation for total least squares is the same as what we consider in this paper: input data are contaminated with noise. Unlike the statistical modeling approach we adopted in this section, the total least squares algorithm is derived from a numerical computation point of view. The resulting formulation is similar to (3), but its solution can be conveniently described by a matrix SVD decomposition. The method has been widely applied in engineering applications, and is known to give better performance than the standard least squares method for problems with uncertain inputs. In our framework, we can regard (3) as the underlying statistical model for total least squares.

For binary classification where $y_i \in \{\pm 1\}$, we consider logistic conditional probability model for $y_i$, while still assume Gaussian noise in the input:

$$p(\mathbf{x}'_i, y_i | \theta) \sim p(\mathbf{x}'_i) \frac{1}{1 + \exp(-\theta^T \mathbf{x}'_i y_i)}, \quad p(\mathbf{x}_i | \theta', \sigma_i, \mathbf{x}'_i) \sim \exp\left( -\frac{\|\mathbf{x}_i - \mathbf{x}'_i\|^2}{2\sigma_i^2} \right).$$

Similar to the total least squares method (3), we obtain the following formulation from (2):

$$\theta = \arg\min_{\theta} \sum_i \inf_{\mathbf{x}'_i} \left[ \ln(1 + e^{-\theta^T \mathbf{x}'_i y_i}) + \frac{\|\mathbf{x}_i - \mathbf{x}'_i\|^2}{2\sigma_i^2} \right]. \tag{4}$$

A well-known disadvantage of logistic model for binary classification is that it does not model deterministic conditional probability (that is, $p(y = 1|\mathbf{x}) = 0, 1$) very well. This problem can be remedied using the support vector machine formulation, which has attractive intuitive geometric interpretations for linearly separable problems. Although in this section a statistical modeling approach is used to gain useful insights, we will focus on support vector machines in the rest of the paper.

## 3  Total support vector classification

Our formulation of support vector classification with uncertain input data is motivated by the total least squares regression method that can be derived from the statistical model (3). We thus call the proposed algorithm total support vector classification (TSVC) algorithm.

We assume that inputs are subject to an additive noise, i.e., $\mathbf{x}'_i = \mathbf{x}_i + \Delta \mathbf{x}_i$ where noise $\Delta \mathbf{x}_i$ follows certain distribution. Bounded and ellipsoidal uncertainties are often discussed in the TLS context [7], and resulting approaches find many real-life applications. Hence instead of assuming Gaussian noise as in (3) and (4), we consider a simple bounded uncertainty model $\|\Delta \mathbf{x}_i\| \leq \delta_i$ with uniform priors. The bound $\delta_i$ has a similar effect of the standard deviation $\sigma_i$ in the Gaussian noise model. However, under the bounded uncertainty model, the squared penalty term $\|\mathbf{x}_i - \mathbf{x}'_i\|^2 / 2\sigma_i^2$ is replaced by a constraint $\|\Delta \mathbf{x}_i\| \leq \delta_i$. Another reason for us to use the bounded uncertainty noise model is that the resulting formulation has a more intuitive geometric interpretation (see Section 4).

SVMs construct classifiers based on separating hyperplanes $\{\mathbf{x} : \mathbf{w}^T \mathbf{x} + b = 0\}$. Hence the parameter $\theta$ in (3) and (4) is replaced by a weight vector $\mathbf{w}$ and a bias $b$. In the separable case, TSVC solves the following problem:

$$\min_{\mathbf{w}, b, \Delta \mathbf{x}_i, i=1,\cdots,\ell} \quad \frac{1}{2} \|\mathbf{w}\|^2$$
$$\text{subject to} \quad y_i \left( \mathbf{w}^T (\mathbf{x}_i + \Delta \mathbf{x}_i) + b \right) \geq 1, \ \|\Delta \mathbf{x}_i\| \leq \delta_i, \ i = 1, \cdots, \ell. \tag{5}$$

For non-separable problems, we follow the standard practice of introducing slack variables $\xi_i$, one for each data point. In the resulting formulation, we simply replace the square loss in (3) or the logistic loss in (4) by the margin-based hinge-loss $\xi = \max\{0, 1 - y(\mathbf{w}^T\mathbf{x} + b)\}$, which is used in the standard SVC.

$$\begin{aligned}
\min_{\mathbf{w}, b, \boldsymbol{\xi}, \Delta\mathbf{x}_i, i=1,\cdots,\ell} \quad & C\sum_{i=1}^{\ell} \xi_i + \frac{1}{2}\|\mathbf{w}\|^2 \\
\text{subject to} \quad & y_i\left(\mathbf{w}^T(\mathbf{x}_i + \Delta\mathbf{x}_i) + b\right) \geq 1 - \xi_i, \; \xi_i \geq 0, \; i = 1, \cdots, \ell, \\
& \|\Delta\mathbf{x}_i\| \leq \delta_i, \; i = 1, \cdots, \ell.
\end{aligned} \quad (6)$$

Note that we introduced the standard Tikhonov regularization term $\frac{1}{2}\|\mathbf{w}\|_2^2$ as usually employed in SVMs. The effect is similar to a Gaussian prior in (3) and (4) with the Bayesian MAP (maximum a posterior) estimator. One can regard (6) as a regularized instance of (2) with a non-probabilistic SVM discriminative loss criterion.

Problems with corrupted inputs are more difficult than problems with no input uncertainty. Even if there is a large margin separator for the original uncorrupted inputs, the observed noisy data may become non-separable. By modifying the noisy input data as in (6), we reconstruct an easier problem, for which we may find a good linear separator. Moreover, by modeling noise in the input data, TSVC becomes less sensitive to data points that are very uncertain since we can find a choice of $\Delta\mathbf{x}_i$ such that $\mathbf{x}_i + \Delta\mathbf{x}_i$ is far from the decision boundary and will not be a support vector. This is illustrated later in Figure 1 (right). TSVC thus constructs classifiers by focusing on the more trust-worthy data that are less uncertain.

## 4 Geometric interpretation

Further investigation reveals an intuitive geometric interpretation for TSVC which allows users to easily grasp the fundamentals of this new formulation. We first derive the following fact that when the optimal $\hat{\mathbf{w}}$ is obtained, the optimal $\Delta\hat{\mathbf{x}}_i$ can be represented in terms of $\hat{\mathbf{w}}$. If $\mathbf{w}$ is fixed in problem (6), optimizing problem (6) is equivalent to minimizing $\sum \xi_i$ over $\Delta\mathbf{x}_i$. The following lemma characterizes the solution.

**Lemma 1.** *For any given hyperplane* $(\mathbf{w}, b)$*, the solution* $\Delta\hat{\mathbf{x}}_i$ *of problem (6) is* $\Delta\hat{\mathbf{x}}_i = y_i\delta_i\frac{\mathbf{w}}{\|\mathbf{w}\|}$, $i = 1, \cdots, \ell$.

**Proof.** Since the noise vector $\Delta\mathbf{x}_i$ only affects $\xi_i$ and does not have impact on other slack variables $\xi_j$, $j \neq i$. The minimization of $\sum \xi_i$ can be decoupled into $\ell$ subproblems of minimizing each $\xi_i = \max\{0, 1 - y_i(\mathbf{w}^T(\mathbf{x}_i + \Delta\mathbf{x}_i) + b)\} = \max\{0, 1 - y_i(\mathbf{w}^T\mathbf{x}_i + b) - y_i\mathbf{w}^T\Delta\mathbf{x}_i\}$ over its corresponding $\Delta\mathbf{x}_i$. By the Cauchy-Schwarz inequality, we have $|y_i\mathbf{w}^T\Delta\mathbf{x}_i| \leq \|\mathbf{w}\| \cdot \|\Delta\mathbf{x}_i\|$ with equality if and only if $\Delta\mathbf{x}_i = c\mathbf{w}$ for some scalar $c$. Since $\Delta\mathbf{x}_i$ is bounded by $\delta_i$, the optimal $\Delta\hat{\mathbf{x}}_i = y_i\delta_i\frac{\mathbf{w}}{\|\mathbf{w}\|}$ and the minimal $\hat{\xi}_i = \max\{0, 1 - y_i(\mathbf{w}^T\mathbf{x}_i + b) - \delta_i\|\mathbf{w}\|\}$. ∎

Define $S_{\mathbf{w}}(\mathbf{X}) = \{\mathbf{x}_i + y_i\delta_i\frac{\mathbf{w}}{\|\mathbf{w}\|}, i = 1, \cdots, \ell\}$. Then $S_{\mathbf{w}}(\mathbf{X})$ is a set of points that are obtained by shifting the original points labeled $+1$ along $\mathbf{w}$ and points labeled $-1$ along $-\mathbf{w}$, respectively, to its individual uncertainty boundary. These shifted points are illustrated in Figure 1(middle) as filled points.

**Theorem 1.** *The optimal hyperplane* $(\hat{\mathbf{w}}, \hat{b})$ *obtained by TSVC (5) separates* $S_{\hat{\mathbf{w}}}(\mathbf{X})$ *with the maximal margin. The optimal hyperplane* $(\hat{\mathbf{w}}, \hat{b})$ *obtained by TSVC (6) separates* $S_{\hat{\mathbf{w}}}(\mathbf{X})$ *with the maximal soft margin.*

**Proof.** 1. If there exists any $\mathbf{w}$ such that $S_{\mathbf{w}}(\mathbf{X})$ is linearly separable, we can solve problem (5) to obtain the largest separation margin. Let $(\hat{\mathbf{w}}, \hat{b}, \Delta\hat{\mathbf{x}}_i)$ be optimal to problem (5). Note that solving problem (5) is equivalent to $\max \; \rho$ subject to constraints $y_i(\mathbf{w}^T(\mathbf{x}_i + $

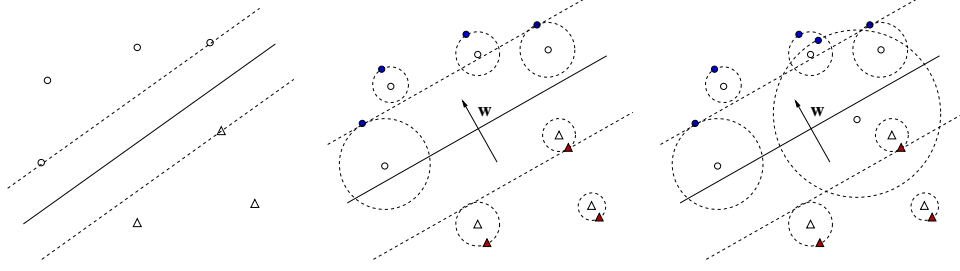

Figure 1: The separating hyperplanes obtained (*left*) by standard SVC and (*middle*) by total SVC (6). The margin can be magnified by taking into account uncertainties. *Right:* TSVC solution is less sensitive to outliers with large noise.

$\Delta \mathbf{x}_i) + b) \geq \rho$ and $\|\mathbf{w}\| = 1$, so the optimal $\rho = \frac{1}{\|\hat{\mathbf{w}}\|}$ [8]. To have the greatest $\rho$, we want to max $y_i(\hat{\mathbf{w}}^T(\mathbf{x}_i + \Delta \mathbf{x}_i) + \hat{b})$ for all $i$'s over $\Delta \mathbf{x}_i$. Hence following similar arguments in Lemma 1, we have $|y_i \hat{\mathbf{w}}^T \Delta \mathbf{x}_i| \leq \|\hat{\mathbf{w}}\| \|\Delta \mathbf{x}_i\| = \delta_i \|\hat{\mathbf{w}}\|$ and when $\Delta \hat{\mathbf{x}}_i = y_i \delta_i \frac{\hat{\mathbf{w}}}{\|\hat{\mathbf{w}}\|}$, the "equal" sign holds.

2. If no $\mathbf{w}$ exists to separate $S_{\mathbf{w}}(\mathbf{X})$ or even when such a $\mathbf{w}$ exists, we may solve problem (6) to achieve the best compromise between the training error and the margin size. Let $\hat{\mathbf{w}}$ be optimal to problem (6). By Lemma 1, the optimal $\Delta \hat{\mathbf{x}}_i = y_i \delta_i \frac{\hat{\mathbf{w}}}{\|\hat{\mathbf{w}}\|}$.

According to the above analysis, we can convert problems (5) and (6) to a problem in variables $\mathbf{w}, b, \boldsymbol{\xi}$, as opposed to optimizing over both $(\mathbf{w}, b, \boldsymbol{\xi})$ and $\Delta \mathbf{x}_i, i = 1, \cdots, \ell$. For example, the linearly non-separable problem (6) becomes

$$\min_{\mathbf{w}, b, \boldsymbol{\xi}} \quad C \sum_{i=1}^{\ell} \xi_i + \frac{1}{2} \|\mathbf{w}\|^2$$
$$\text{subject to} \quad y_i \left( \mathbf{w}^T \mathbf{x}_i + b \right) + \delta_i \|\mathbf{w}\| \geq 1 - \xi_i, \ \xi_i \geq 0, \ i = 1, \cdots, \ell. \tag{7}$$

Solving problem (7) yields an optimal solution to problem (6), and problem (7) can be interpreted as finding $(\mathbf{w}, b)$ to separate $S_{\mathbf{w}}(\mathbf{X})$ with the maximal soft margin. The similar argument holds true for the linearly separable case. ∎

## 5   Solving and kernelizing TSVC

TSVC problem (6) can be recast to a second-order cone program (SOCP) as usually done in TLS or Robust LS methods [7, 4]. However, directly implementing this SOCP will be computationally quite expensive. Moreover, the SOCP formulation involves a large amount of redundant variables, so a typical SOCP solver will take much longer time to achieve an optimal solution. We propose a simple iterative approach as follows based on alternating optimization method [1].

**Algorithm 1**
Initialize $\Delta \mathbf{x}_i = 0$, repeat the following two steps until a termination criterion is met:
1. Fix $\Delta \mathbf{x}_i, \ i = 1, \cdots, \ell$ to the current value, solve problem (6) for $\mathbf{w}$, $b$, and $\boldsymbol{\xi}$.
2. Fix $\mathbf{w}$, $b$ to the current value, solve problem (6) for $\Delta \mathbf{x}_i, i = 1, \cdots, \ell$, and $\boldsymbol{\xi}$.

The first step of Algorithm 1 solves no more than a standard SVM by treating $\mathbf{x}_i + \Delta \mathbf{x}_i$ as the training examples. Similar to how SVMs are usually optimized, we can solve the dual SVM formulation [8] for $\hat{\mathbf{w}}, \hat{b}$. The second step of Algorithm 1 solves a problem which has been discussed in Lemma 1. No optimization solver is needed. The solution $\Delta \mathbf{x}_i$ of the second step has a closed form in terms of the fixed $\mathbf{w}$.

## 5.1 TSVC with linear functions

When only linear functions are considered, an alternative exists to solve problem (6) other than Algorithm 1. As analyzed in [5, 3], Tikhonov regularization $\min\ C \sum \xi_i + \frac{1}{2}\|\mathbf{w}\|^2$ has an important equivalent formulation as $\min \sum \xi_i$, subject to $\|\mathbf{w}\| \leq \gamma$ where $\gamma$ is a positive constant. It can be shown that if $\gamma \leq \|\mathbf{w}^*\|$ where $\mathbf{w}^*$ is the solution to problem (6) with $\frac{1}{2}\|\mathbf{w}\|^2$ removed, then the solution for the constraint problem is identical to the solution of the Tikhonov regularization problem for an appropriately chosen $C$. Furthermore, at optimality, the constraint $\|\mathbf{w}\| \leq \gamma$ is active, which means $\|\hat{\mathbf{w}}\| = \gamma$. Hence TSVC problem (7) can be converted to a simple SOCP with the constraint $\|\mathbf{w}\| \leq \gamma$ or a quadratically constrained quadratic program (QCQP) as follows if equivalently using $\|\mathbf{w}\|^2 \leq \gamma^2$.

$$\begin{aligned}
&\min_{\mathbf{w},b,\boldsymbol{\xi}} && \sum_{i=1}^{\ell} \xi_i \\
&\text{subject to} && y_i\left(\mathbf{w}^T\mathbf{x}_i + b\right) + \gamma\delta_i \geq 1 - \xi_i, \ \xi_i \geq 0, \ i = 1,\cdots,\ell, \quad \|\mathbf{w}\|^2 \leq \gamma^2.
\end{aligned} \tag{8}$$

This QCQP produces exactly the same solution as problem (6) but is much easier to implement than (6) since it contains much less variables. By duality analysis similarly adopted in [3], problem (8) has a dual formulation in dual variables $\boldsymbol{\alpha}$ as follows

$$\begin{aligned}
&\min_{\boldsymbol{\alpha}} && \gamma\sqrt{\sum_{i,j=1}^{\ell} \alpha_i\alpha_j y_i y_j \mathbf{x}_i^T\mathbf{x}_j} - \sum_{i=1}^{\ell}(1 - \gamma\delta_i)\alpha_i \\
&\text{subject to} && \sum_{i=1}^{\ell} \alpha_i y_i = 0, \quad 0 \leq \alpha_i \leq 1, \ i = 1,\cdots,\ell.
\end{aligned} \tag{9}$$

## 5.2 TSVC with kernels

By using a kernel function $k$, the input vector $\mathbf{x}_i$ is mapped to $\Phi(\mathbf{x}_i)$ in a usually high dimensional feature space. The uncertainty in the input data introduces uncertainties for images $\Phi(\mathbf{x}_i)$ in the feature space. TSVC can be generalized to construct separating hyperplanes in the feature space using the images of input vectors and the mapped uncertainties. One possible generalization of TSVC is to assume the images are still subject to an additive noise and the uncertainty model in the feature space can be represented as $\|\Delta\Phi(\mathbf{x}_i)\| \leq \delta_i$. Then following the similar analysis in Sections 4 and 5.1, we obtain a problem same as (8) only with $\mathbf{x}_i$ replaced by $\Phi(\mathbf{x}_i)$ and $\Delta\mathbf{x}_i$ replaced by $\Delta\Phi(\mathbf{x}_i)$, which can be easily kernelized by solving its dual formulation (9) with inner products $\mathbf{x}_i^T\mathbf{x}_j$ replaced by $k(\mathbf{x}_i,\mathbf{x}_j)$.

It is more realistic, however, that we are only able to estimate uncertainties in the input space as bounded spheres $\|\Delta\mathbf{x}_i\| \leq \delta_i$. When the uncertainty sphere is mapped to the feature space, the mapped uncertainty region may correspond to an irregular shape in the feature space, which brings difficulties to the optimization of TSVC. We thus propose an approximation strategy for Algorithm 1 based on the first order Taylor expansion of $k$.

A kernel function $k(\mathbf{x},\mathbf{z})$ takes two arguments $\mathbf{x}$ and $\mathbf{z}$. When we fix one of the arguments, for example $\mathbf{z}$, $k$ can be viewed as a function of the other argument $\mathbf{x}$. The first order Taylor expansion of $k$ with respect to $\mathbf{x}$ is $k(\mathbf{x}_i + \Delta\mathbf{x},\cdot) = k(\mathbf{x}_i,\cdot) + \Delta\mathbf{x}^T k'(\mathbf{x}_i,\cdot)$ where $k'(\mathbf{x}_i,\cdot)$ is the gradient of $k$ with respect to $\mathbf{x}$ at point $\mathbf{x}_i$.

Solving the dual SVM formulation in step 1 of Algorithm 1 with $\Delta\mathbf{x}_j$ fixed to $\Delta\bar{\mathbf{x}}_j$ yields a solution $(\bar{\mathbf{w}} = \sum_j y_j\bar{\alpha}_j\Phi(\mathbf{x}_j + \Delta\bar{\mathbf{x}}_j),\bar{b})$ and thus a predictor $f(\mathbf{x}) = \sum_j y_j\bar{\alpha}_j k(\mathbf{x},\mathbf{x}_j + \Delta\bar{\mathbf{x}}_j) + \bar{b}$. In step 2, we set $(\mathbf{w},b)$ to $(\bar{\mathbf{w}},\bar{b})$ and minimize $\sum \xi_i$ over $\Delta\mathbf{x}_i$, which as we discussed in Lemma 1, amounts to minimizing each $\xi_i = \max\{0, 1 - y_i(\sum_j y_j\bar{\alpha}_j k(\mathbf{x}_i + \Delta\mathbf{x}_i,\mathbf{x}_j + \Delta\bar{\mathbf{x}}_j) + b)\}$ over $\Delta\mathbf{x}_i$. Applying the Taylor expansion yields

$$\begin{aligned}
&y_i\left(\sum_j y_j\bar{\alpha}_j k(\mathbf{x}_i + \Delta\mathbf{x}_i,\mathbf{x}_j + \Delta\bar{\mathbf{x}}_j) + b\right) \\
=\ &y_i\left(\sum_j y_j\bar{\alpha}_j k(\mathbf{x}_i,\mathbf{x}_j + \Delta\bar{\mathbf{x}}_j) + b\right) + y_i\Delta\mathbf{x}_i^T \sum_j y_j\bar{\alpha}_j k'(\mathbf{x}_i,\mathbf{x}_j + \Delta\bar{\mathbf{x}}_j).
\end{aligned}$$

Table 1: Average test error percentages of TSVC and standard SVC algorithms on synthetic problems (*left and middle* ) and digits classification problems (*right*).

| $\ell$ | Synthetic linear target | | | | | Synthetic quadratic target | | | | | Digits | |
|---|---|---|---|---|---|---|---|---|---|---|---|---|
| | 20 | 30 | 50 | 100 | 150 | 20 | 30 | 50 | 100 | 150 | 100 | 500 |
| SVC | 8.9 | 7.8 | 5.5 | 2.9 | 2.1 | 9.9 | 7.5 | 6.7 | 3.2 | 2.8 | 24.35 | 18.91 |
| TSVC | 6.1 | 5.2 | 3.8 | 2.1 | 1.6 | 7.9 | 6.1 | 4.4 | 2.8 | 2.4 | 23.00 | 16.10 |

The optimal $\Delta \mathbf{x}_i = y_i \delta_i \frac{\mathbf{v}_i}{\|\mathbf{v}_i\|}$ where $\mathbf{v}_i = \sum y_j \bar{\alpha}_j k'(\mathbf{x}_i, \mathbf{x}_j + \Delta \bar{\mathbf{x}}_j)$ by the Cauchy-Schwarz inequality. A closed-form approximate solution for the second step is thus acquired.

## 6   Experiments

Two sets of simulations were performed, one on synthetic datasets and one on NIST handwritten digits, to validate the proposed TSVC algorithm. We used the commercial optimization package ILOG CPLEX 9.0 to solve problems (8), (9) and the standard SVC dual problem that is part of Algorithm 1.

In the experiments with synthetic data in 2 dimensions, we generated $\ell$ (=20, 30, 50, 100, 150) training examples $\mathbf{x}_i$ from the uniform distribution on $[-5, 5]^2$. Two binary classification problems were created with target separating functions as $x_1 - x_2 = 0$ and $x_1^2 + x_2^2 = 9$, respectively. We used TSVC with linear functions for the first problem and TSVC with the quadratic kernel $(\mathbf{x}_i^T \mathbf{x}_j)^2$ for the second problem. The input vectors $\mathbf{x}_i$ were contaminated by Gaussian noise with mean [0,0] and covariance matrix $\Sigma = \sigma_i \mathbf{I}$ where $\sigma_i$ was randomly chosen from $[0.1, 0.8]$. The matrix $\mathbf{I}$ denotes the $2 \times 2$ identity matrix. To produce an outlier effect, we randomly chose $0.1\ell$ examples from the first $0.2\ell$ examples after examples were ordered in an ascending order of their distances to the target boundary. For these $0.1\ell$ examples, noise was generated using a larger $\sigma$ randomly drawn from $[0.5, 2]$. Models obtained by the standard SVC and TSVC were tested on a test set of 10000 examples that were generated from the same distribution and target functions but without contamination. We performed 50 trials for each experimental setting. The misclassification error rates averaged over the 50 trials are reported in Table 1. TSVC performed overall better than SVC. Two representative modeling results of $\ell = 50$ are also visually depicted in Figure 2.

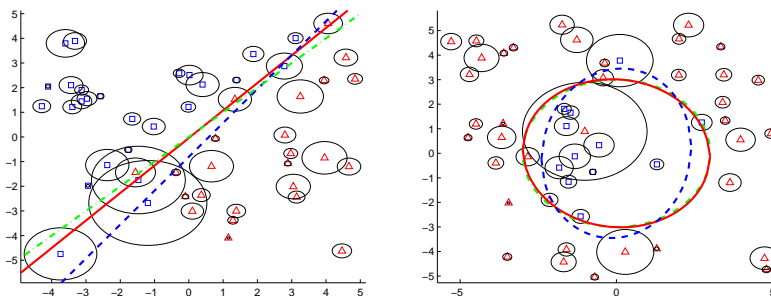

Figure 2: Results obtained by TSVC (solid lines) and standard SVC (dash lines) for the problem with (*left*) a linear target function and the problem with (*right*) a quadratic target function. The true target functions are illustrated using dash-dot lines.

The NIST database of handwritten digits does not contain any uncertainty information

originally. We created uncertainties by image distortions. Different types of distortions can present in real-life data. We simulated it only by rotating images. We used $\ell$ (=100, 500) digits from the beginning of the database in training and 2000 digits from the end of the database in test. We discriminated between odd numbers and even numbers. The angle of rotation for each digit was randomly chosen from $[-8^o, 8^o]$. The uncertainty upper bounds $\delta_i$ can be regarded as tuning parameters. We simply set all $\delta_i = \delta$. The data was preprocessed in the following way: training examples were centered to have mean 0 and scaled to have standard deviation 1. The test data was preprocessed using the mean and standard deviation of training examples. We performed 50 trials with TSVC and SVC using the linear kernel, which means we need to solve problem (9). Results are reported in Table 1 and the tuned parameter $\delta$ was 1.38 for $\ell = 100$ and 1.43 for $\ell = 500$. We conjecture that TSVC performance can be further improved if we obtain an estimate of $\delta_i$.

## 7    Discussions

We investigated a new learning model in which the observed input is corrupted with noise. Based on a probability modeling approach, we derived a general statistical formulation where unobserved input is modeled as a hidden mixture component. Under this framework, we were able to develop estimation methods that take input uncertainty into consideration. Motivated by this probability modeling approach, we proposed a new SVM classification formulation that handles input uncertainty. This formulation has an intuitive geometric interpretation. Moreover, we presented simple numerical algorithms which can be used to solve the resulting formulation efficiently. Two empirical examples, one artificial and one with real data, were used to illustrate that the new method is superior to the standard SVM for problems with noisy input data. A related approach, with a different focus, is presented in [2]. Our work attempts to recover the original classifier from the corrupted training data, and hence we evaluated the performance on clean test data. In our statistical modeling framework, rigorously speaking, the input uncertainty of test-data should be handled by a mixture model (or a voted classifier under the noisy input distribution). The formulation in [2] was designed to separate the training data under the worst input noise configuration instead of the most likely configuration in our case. The purpose is to directly handle test input uncertainty with a single linear classifier under the worst possible error setting. The relationship and advantages of these different approaches require further investigation.

## References

[1] J. Bezdek and R. Hathaway. Convergence of alternating optimization. *Neural, Parallel Sci. Comput.*, 11:351–368, 2003.

[2] C. Bhattacharyya, K.S. Pannagadatta, and A. J. Smola. A second order cone programming formulation for classifying missing data. In *NIPS, Vol 17*, 2005.

[3] J. Bi and V. N. Vapnik. Learning with rigorous support vector machines. In M. Warmuth and B. Schölkopf, editors, *Proceedings of the 16th Annual Conference on Learning Theory*, pages 35–42, Menlo Park, CA, 2003. AAAI Press.

[4] L. El Ghaoui and H. Lebret. Robust solutions to least-squares problems with uncertain data. *SIAM Journal on Matrix Analysis and Applications*, 18:1035–1064, 1997.

[5] G. H. Golub, P. C. Hansen, and D. P. O'Leary. Tikhonov regularization and total least squares. *SIAM Journal on Numerical Analysis*, 30:185–194, 1999.

[6] G. H. Golub and C. F. Van Loan. An analysis of the total least squares problem. *SIAM Journal on Numerical Analysis*, 17:883–893, 1980.

[7] S. Van Huffel and J. Vandewalle. *The Total Least Squares Problem: Computational Aspects and Analysis, in Frontiers in Applied Mathematics 9*. SIAM Press, Philadelphia, PA, 1991.

[8] V. N. Vapnik. *Statistical Learning Theory*. John Wiley & Sons, Inc., New York, 1998.
